# Linear Response for Approximate Inference

**Max Welling**
Department of Computer Science
University of Toronto
Toronto M5S 3G4 Canada
welling@cs.utoronto.ca

**Yee Whye Teh**
Computer Science Division
University of California at Berkeley
Berkeley CA94720 USA
ywteh@eecs.berkeley.edu

## Abstract

Belief propagation on cyclic graphs is an efficient algorithm for computing approximate marginal probability distributions over single nodes and neighboring nodes in the graph. In this paper we propose two new algorithms for approximating joint probabilities of arbitrary pairs of nodes and prove a number of desirable properties that these estimates fulfill. The first algorithm is a propagation algorithm which is shown to converge if belief propagation converges to a stable fixed point. The second algorithm is based on matrix inversion. Experiments compare a number of competing methods.

## 1 Introduction

Belief propagation (BP) has become an important tool for approximate inference on graphs with cycles. Especially in the field of "error correction decoding", it has brought performance very close to the Shannon limit. BP was studied in a number of papers which have gradually increased our understanding of the convergence properties and accuracy of the algorithm. In particular, recent developments show that the stable fixed points are local minima of the Bethe free energy [10, 1], which paved the way for more accurate "generalized belief propagation" algorithms and convergent alternatives to BP [11, 6].

Despite its success, BP does not provide a prescription to compute joint probabilities over pairs of non-neighboring nodes in the graph. When the graph is a tree, there is a single chain connecting any two nodes, and dynamic programming can be used to efficiently integrate out the internal variables. However, when cycles exist, it is not clear what approximate procedure is appropriate. It is precisely this problem that we will address in this paper. We show that the required estimates can be obtained by computing the "sensitivity" of the node marginals to small changes in the node potentials. Based on this idea, we present two algorithms to estimate the joint probabilities of arbitrary pairs of nodes.

These results are interesting in the inference domain but may also have future applications to learning graphical models from data. For instance, information about dependencies between random variables is relevant for learning the structure of a graph and the parameters encoding the interactions.

## 2 Belief Propagation on Factor Graphs

Let $V$ index a collection of random variables $\{X_i\}_{i \in V}$ and let $x_i$ denote values of $X_i$. For a subset of nodes $\alpha \subset V$ let $X_\alpha = \{X_i\}_{i \in \alpha}$ be the variable associated with that subset, and $x_\alpha$ be values of $X_\alpha$. Let $A$ be a family of such subsets of $V$. The probability distribution over $X \doteq X_V$ is assumed to have the following form,

$$P_X(X = x) = \frac{1}{Z} \prod_{\alpha \in A} \psi_\alpha(x_\alpha) \prod_{i \in V} \psi_i(x_i) \tag{1}$$

where $Z$ is the normalization constant (the partition function) and $\psi_\alpha, \psi_i$ are positive potential functions defined on subsets and single nodes respectively. In the following we will write $P(x) \doteq P_X(X = x)$ for notational simplicity. The decomposition of (1) is consistent with a factor graph with function nodes over $X_\alpha$ and variables nodes $X_i$. For each $i \in V$ denote its neighbors by $\mathcal{N}_i = \{\alpha \in A : \alpha \ni i\}$ and for each subset $\alpha$ its neighbors are simply $\mathcal{N}_\alpha = \{i \in \alpha\}$.

Factor graphs are a convenient representation for structured probabilistic models and subsume undirected graphical models and acyclic directed graphical models [3]. Further, there is a simple message passing algorithm for approximate inference that generalizes the belief propagation algorithms on both undirected and acyclic directed graphical models,

$$n_{i\alpha}(x_i) \leftarrow \psi_i(x_i) \prod_{\beta \in \mathcal{N}_i \setminus \alpha} m_{\beta i}(x_i) \qquad m_{\alpha i}(x_i) \leftarrow \sum_{x_{\alpha \setminus i}} \psi_\alpha(x_\alpha) \prod_{j \in \mathcal{N}_\alpha \setminus i} n_{j\alpha}(x_j) \tag{2}$$

where $n_{i\alpha}(x_i)$ represents a message from variable node $i$ to factor node $\alpha$ and vice versa for message $m_{\alpha i}(x_i)$. Marginal distributions over factor nodes and variable nodes are expressed in terms of these messages as follows,

$$b_\alpha(x_\alpha) = \frac{1}{\gamma_\alpha} \psi_\alpha(x_\alpha) \prod_{i \in \mathcal{N}_\alpha} n_{i\alpha}(x_i) \qquad b_i(x_i) = \frac{1}{\gamma_i} \psi_i(x_i) \prod_{\alpha \in \mathcal{N}_i} m_{\alpha i}(x_i) \tag{3}$$

where $\gamma_i, \gamma_\alpha$ are normalization constants. It was recently established in [10, 1] that stable fixed points of these update equations correspond to local minima of the Bethe-Gibbs free energy given by,

$$G^{\text{BP}}(\{b_i^{\text{BP}}, b_\alpha^{\text{BP}}\}) = \sum_\alpha \sum_{x_\alpha} b_\alpha^{\text{BP}}(x_\alpha) \log \frac{b_\alpha^{\text{BP}}(x_\alpha)}{\psi_\alpha(x_\alpha)} + \sum_i \sum_{x_i} b_i^{\text{BP}}(x_i) \log \frac{b_i^{\text{BP}}(x_i)^{c_i}}{\psi_i(x_i)} \tag{4}$$

with $c_i = 1 - |\mathcal{N}_i|$ and the marginals are subject to the following *local* constraints:

$$\sum_{x_{\alpha \setminus i}} b_\alpha^{\text{BP}}(x_\alpha) = b_i^{\text{BP}}(x_i), \qquad \sum_{x_\alpha} b_\alpha(x_\alpha) = 1, \qquad \forall \alpha \in A, \ i \in \alpha \tag{5}$$

Since only local constraints are enforced it is no longer guaranteed that the set of marginals $\{b_i^{\text{BP}}, b_\alpha^{\text{BP}}\}$ are consistent with a single joint distribution $B(x)$.

## 3 Linear Response

In the following we will be interested in computing estimates of joint probability distributions for arbitrary pairs of nodes. We propose a method based on the linear response theorem. The idea is to study changes in the system when we perturb single node potentials,

$$\log \psi_i(x_i) = \log \psi_i^0(x_i) + \theta_i(x_i) \tag{6}$$

The superscript $^0$ indicates unperturbed quantities in (6) and the following. Let $\theta = \{\theta_i\}$ and define the cumulant generating function of $P(X)$ (up to a constant) as,

$$F(\theta) = -\log \sum_x \prod_{\alpha \in A} \psi_\alpha(x_\alpha) \prod_{i \in V} \psi_i^0(x_i) e^{\theta_i(x_i)} \tag{7}$$

Differentiating $F(\theta)$ with respect to $\theta$ gives the cumulants of $P(x)$,

$$-\left.\frac{\partial F(\theta)}{\partial \theta_j(x_j)}\right|_{\theta=0} = p_j(x_j) \tag{8}$$

$$-\left.\frac{\partial^2 F(\theta)}{\partial \theta_i(x_i) \partial \theta_j(x_j)}\right|_{\theta=0} = \left.\frac{\partial p_j(x_j)}{\partial \theta_i(x_i)}\right|_{\theta=0} = \begin{cases} p_{ij}(x_i, x_j) - p_i(x_i)p_j(x_j) & \text{if } i \neq j \\ p_i(x_i)\delta_{x_i,x_j} - p_i(x_i)p_j(x_j) & \text{if } i = j \end{cases} \tag{9}$$

where $p_i, p_{ij}$ are single and pairwise marginals of $P(x)$. Expressions for higher order cumulants can be derived by taking further derivatives of $-F(\theta)$.

Notice from (9) that the covariance estimates are obtained by studying the perturbations in $p_j(x_j)$ as we vary $\theta_i(x_i)$. This is not practical in general since calculating $p_j(x_j)$ itself is intractable. Instead, we consider perturbations of approximate marginal distributions $\{b_j\}$. In the following we will assume that $b_j(x_j; \theta)$ (with the dependence on $\theta$ made explicit) are the beliefs at a local minimum of the BP-Gibbs free energy (subject to constraints).

In analogy to (9), let $C_{ij}(x_i, x_j) = \left.\frac{\partial b_j(x_j; \theta)}{\partial \theta_i(x_i)}\right|_{\theta=0}$ be the linear response estimated covariance, and define the linear response estimated joint pairwise marginal as

$$b_{ij}^{\text{LR}}(x_i, x_j) = b_i^0(x_i)b_j^0(x_j) + C_{ij}(x_i, x_j) \tag{10}$$

where $b_i^0(x_i) \doteq b_i(x_i; \theta = 0)$. We will show that $b_{ij}^{\text{LR}}$ and $C_{ij}$ satisfy a number of important properties which make them suitable as approximations of joint marginals and covariances.

First we show that $C_{ij}(x_i, x_j)$ can be interpreted as the Hessian of a well-behaved convex function. Let $\mathcal{C}$ be the set of beliefs that satisfy the constraints (5). The approximate marginals $\{b_i^0\}$ along with the joint marginals $\{b_\alpha^0\}$ form a local minimum of the Bethe-Gibbs free energy (subject to $b^0 \doteq \{b_i^0, b_\alpha^0\} \in \mathcal{C}$). Assume that $b^0$ is a *strict* local minimum of $G^{\text{BP}}$ (the strict local minimality is in fact attained if we use loopy belief propagation [1]). That is, there is an open domain $\mathcal{D}$ containing $b^0$ such that $G^{\text{BP}}(b^0) < G^{\text{BP}}(b)$ for each $b \in \mathcal{D} \cap \mathcal{C}\backslash b^0$. Now we can define

$$G^*(\theta) = \inf_{b \in \mathcal{D} \cap \mathcal{C}} G^{\text{BP}}(b) - \sum_{i,x_i} b_i(x_i)\theta_i(x_i) \tag{11}$$

$G^*(\theta)$ is a concave function since it is the infimum of a set of linear functions in $\theta$. Further $G^*(0) = G^{\text{BP}}(b^0)$. Since $b^0$ is a strict local minimum when $\theta = 0$, small perturbations in $\theta$ will result in small perturbations in $b^0$, so that $G^*$ is well-behaved on an open neighborhood around $\theta = 0$. Differentiating $G^*(\theta)$, we get $\frac{\partial G^*(\theta)}{\partial \theta_j(x_j)} = -b_j(x_j; \theta)$ so we now have

$$C_{ij}(x_i, x_j) = \left.\frac{\partial b_j(x_j; \theta)}{\partial \theta_i(x_i)}\right|_{\theta=0} = -\left.\frac{\partial^2 G^*(\theta)}{\partial \theta_i(x_i) \partial \theta_j(x_j)}\right|_{\theta=0} \tag{12}$$

In essence, we can interpret $G^*(\theta)$ as a *local* convex dual of $G^{\text{BP}}(b)$ (by restricting attention to $\mathcal{D}$). Since $G^{\text{BP}}$ is an approximation to the exact Gibbs free energy [8], which is in turn dual to $F(\theta)$ [4], $G^*(\theta)$ can be seen as an approximation to $F(\theta)$ for small values of $\theta$. For that reason we can take its second derivatives $C_{ij}(x_i, x_j)$ as approximations to the exact covariances (which are second derivatives of $-F(\theta)$).

**Theorem 1** *The approximate covariance satisfies the following symmetry:*

$$C_{ij}(x_i, x_j) = C_{ji}(x_j, x_i) \tag{13}$$

**Proof:** The covariances are second derivatives of $-G^*(\theta)$ at $\theta = 0$ so we can interchange the order of the derivatives since $G^*(\theta)$ is well-behaved on a neighborhood around $\theta = 0$. $\square$

**Theorem 2** *The approximate covariance satisfies the following "marginalization" conditions for each $x_i, x_j$:*

$$\sum_{x_i'} C_{ij}(x_i', x_j) = \sum_{x_j'} C_{ij}(x_i, x_j') = 0 \tag{14}$$

*As a result the approximate joint marginals satisfy local marginalization constraints:*

$$\sum_{x_i'} b_{ij}^{\mathrm{LR}}(x_i', x_j) = b_j^0(x_j) \qquad\qquad \sum_{x_j'} b_{ij}^{\mathrm{LR}}(x_i, x_j') = b_i^0(x_i) \tag{15}$$

**Proof:** Using the definition of $C_{ij}(x_i, x_j)$ and marginalization constraints for $b_j^0$,

$$\sum_{x_j'} C_{ij}(x_i, x_j') = \sum_{x_j'} \frac{\partial b_j(x_j'; \theta)}{\partial \theta_i(x_i)}\Big|_{\theta=0} = \frac{\partial \sum_{x_j'} b_j(x_j'; \theta)}{\partial \theta_i(x_i)}\Big|_{\theta=0} = \frac{\partial}{\partial \theta_i(x_i)} 1\Big|_{\theta=0} = 0 \tag{16}$$

The constraint $\sum_{x_i'} C_{ij}(x_i', x_j) = 0$ follows from the symmetry (13), while the corresponding marginalization (15) follows from (14) and the definition of $b_{ij}^{\mathrm{LR}}$. $\square$

Since $-F(\theta)$ is convex, its Hessian matrix with entries given in (9) is positive semi-definite. Similarly, since the approximate covariances $C_{ij}(x_i, x_j)$ are second derivatives of a convex function $-G^*(\theta)$, we have:

**Theorem 3** *The matrix formed from the approximate covariances $C_{ij}(x_i, x_j)$ by varying $i$ and $x_i$ over the rows and varying $j, x_j$ over the columns is positive semi-definite.*

Using the above results we can reinterpret the linear response correction as a "projection" of the (only locally consistent) beliefs $\{b_i^0, b_\alpha^0\}$ onto a set of beliefs $\{b_i^0, b_{ij}^{\mathrm{LR}}\}$ that is both locally consistent (theorem 2) and satisfies the global constraint of being positive semi-definite (theorem 3)[1].

## 4 Propagating Perturbations for Linear Response

Recall from (10) that we need the first derivative of $b_i(x_i; \theta)$ with respect to $\theta_j(x_j)$ at $\theta = 0$. This does not automatically imply that we need an analytic expression for $b_i(x_i; \theta)$ in terms of $\theta$. In this section we show how we may compute these first derivatives by expanding all quantities and equations *up to first order* in $\theta$ and keeping track of first order dependencies.

First we assume that belief propagation has converged to a stable fixed point. We expand the beliefs and messages up to first order as[2]

$$b_i(x_i; \theta) = b_i^0(x_i)\left(1 + \sum_{j, y_j} R_{ij}(x_i, y_j)\theta_j(y_j)\right) \tag{17}$$

$$n_{i\alpha}(x_i) = n_{i\alpha}^0(x_i)\left(1 + \sum_{k, y_k} N_{i\alpha,k}(x_i, y_k)\theta_k(y_k)\right) \tag{18}$$

$$m_{\alpha i}(x_i) = m_{\alpha i}^0(x_i)\left(1 + \sum_{k, y_k} M_{\alpha i,k}(x_i, y_k)\theta_k(y_k)\right) \tag{19}$$

The "response matrices" $R_{ij}, N_{i\alpha,j}, M_{\alpha i,j}$ measure the sensitivities of the corresponding logarithms of beliefs and messages to changes in the log potentials $\log \psi_j(y_j)$ at node $j$.

Next, inserting the expansions (6,18,19) into the belief propagation equations (2) and matching first order terms, we arrive at the following update equations for the "super-messages" $M_{\alpha i,k}(x_i, y_k)$ and $N_{i\alpha,k}(x_i, y_k)$,

$$N_{i\alpha,k}(x_i, y_k) \leftarrow \delta_{ik}\delta_{x_i y_k} + \sum_{\beta \in \mathcal{N}_i \backslash \alpha} M_{\beta i,k}(x_i, y_k) \tag{20}$$

$$M_{\alpha i,k}(x_i, y_k) \leftarrow \sum_{x_{\alpha \backslash i}} \frac{\psi_\alpha(x_\alpha)}{m_{\alpha i}^0(x_i)} \prod_{j \in \mathcal{N}_\alpha \backslash i} n_{j\alpha}^0(x_j) \sum_{j \in \mathcal{N}_\alpha \backslash i} N_{j\alpha,k}(x_j, y_k) \tag{21}$$

The super-messages are initialized at $M_{\alpha i,k} = N_{i\alpha,k} = 0$ and updated using (20,21) until convergence. Just as for belief propagation, where messages are normalized to avoid numerical over or under flow, after each update the super-messages are "normalized" as follows,

$$M_{\alpha i,k}(x_i, y_k) \leftarrow M_{\alpha i,k}(x_i, y_k) - \sum_{x_i} M_{\alpha i,k}(x_i, y_k) \tag{22}$$

and similarly for $N_{i\alpha,k}$. After the above fixed point equations have converged, we compute the response matrix $R_{ij}(x_i, x_j)$ by again inserting the expansions (6,17,19) into (3) and matching first order terms,

$$R_{ij}(x_i, x_j) = \delta_{ij}\delta_{x_i x_j} + \sum_{\alpha \in \mathcal{N}_i} M_{\alpha i,j}(x_i, x_j) \tag{23}$$

The constraints (14) (which follow from the normalization of $b_i(x_i; \theta)$ and $b_i^0(x_i)$) translate into $\sum_{x_i} b_i^0(x_i) R_{ij}(x_i, y_j) = 0$ and it is not hard to verify that the following shift can be applied to accomplish this,

$$R_{ij}(x_i, y_j) \leftarrow R_{ij}(x_i, y_j) - \sum_{x_i} b_i^0(x_i) R_{ij}(x_i, y_j) \tag{24}$$

Finally, combining (17) with (12), we get

$$C_{ij}(x_i, x_j) = b_i^0(x_i) R_{ij}(x_i, x_j) \tag{25}$$

**Theorem 4** *If the factor graph has no loops then the linear response estimates defined in (25) are exact. Moreover, there exists a scheduling of the super-messages such that the algorithm converges after just one iteration (i.e. every message is updated just once).*

**Sketch of Proof:** Both results follow from the fact that belief propagation on tree structured factor graphs computes the exact single node marginals for arbitrary $\theta$. Since the super-messages are the first order terms of the BP updates with arbitrary $\theta$, we can invoke the exact linear response theorem given by (8) and (9) to claim that the algorithm converges to the exact joint pairwise marginal distributions. $\square$

For graphs with cycles, BP is not guaranteed to converge. We can however still prove the following strong result.

**Theorem 5** *If the messages $\{m_{\alpha i}^0(x_i), n_{i\alpha}^0(x_i)\}$ have converged to a stable fixed point, then the update equations for the super-messages (20,21,22) will also converge to a unique stable fixed point, using any scheduling of the super-messages.*

**Sketch of Proof**[3]: We first note that the updates (20,21,22) form a linear system of equations which can only have one stable fixed point. The existence and stability of this fixed

point is proven by observing that the first order term is identical to the one obtained from a linear expansion of the BP equations (2) around its stable fixed point. Finally, the Stein-Rosenberg theorem guarantees that any scheduling will converge to the same fixed point. □

## 5   Inverting Matrices for Linear Response

In this section we describe an alternative method to compute $\frac{\partial b_i(x_i)}{\partial \theta_k(x_k)}$ by first computing $\frac{\partial \theta_i(x_i)}{\partial b_k(x_k)}$ and then inverting the matrix formed by flattened $\{i, x_i\}$ into a row index and $\{k, x_k\}$ into a column index. This method is a direct extension of [2]. The intuition is that while perturbations in a single $\theta_i(x_i)$ affect the whole system, perturbations in a single $b_i(x_i)$ (while keeping the others fixed) affect each subsystem $\alpha \in A$ *independently* (see [8]). This makes it easier to compute $\frac{\partial \theta_i(x_i)}{\partial b_k(x_k)}$ then to compute $\frac{\partial b_i(x_i)}{\partial \theta_k(x_k)}$.

First we propose minimal representations for $b_i$, $\theta_i$ and the messages. We assume that for each node $i$ there is a distinguished value $x_i = 0$. Set $\theta_i(0) = 0$ while functionally define $b_i(0) = 1 - \sum_{x_i \neq 0} b_i(x_i)$. Now the matrix formed by $\frac{\partial \theta_i(x_i)}{\partial b_k(x_k)}$ for each $i$, $k$ *and* $x_i, x_k \neq 0$ is invertible and its inverse gives us the desired covariances for $x_i, x_k \neq 0$. Values for $x_i = 0$ or $x_k = 0$ can then be computed using (14). We will also need minimal representations for the messages. This can be achieved by defining new quantities $\lambda_{i\alpha}(x_i) = \log \frac{n_{i\alpha}(x_i)}{n_{i\alpha}(0)}$ for all $i$ and $x_i \neq 0$. The $\lambda_{i\alpha}$'s can be interpreted as Lagrange multipliers to enforce the consistency constraints (5) [10]. We will use these multipliers instead of the messages in this section.

Re-expressing the fixed point equations (2,3) in terms of $b_i$'s and $\lambda_{i\alpha}$'s only, and introducing the perturbations $\theta_i$, we get:

$$\left(\frac{b_i(x_i)}{b_i(0)}\right)^{c_i} = \frac{\psi_i(x_i)}{\psi_i(0)} e^{\theta_i(x_i)} \prod_{\alpha \in \mathcal{N}_i} e^{-\lambda_{i\alpha}(x_i)} \qquad \text{for all } i, x_i \neq 0 \qquad (26)$$

$$b_i(x_i) = \frac{\sum_{x_{\alpha \setminus i}} \psi_\alpha(x_\alpha) \prod_{j \in \mathcal{N}_\alpha} e^{\lambda_{j\alpha}(x_j)}}{\sum_{x_\alpha} \psi_\alpha(x_\alpha) \prod_{j \in \mathcal{N}_\alpha} e^{\lambda_{j\alpha}(x_j)}} \qquad \text{for all } i, \alpha \in \mathcal{N}_i, x_i \neq 0 \qquad (27)$$

Differentiating the logarithm of (26) with respect to $b_k(x_k)$, we get

$$\frac{\partial \theta_i(x_i)}{\partial b_k(x_k)} = c_i \delta_{ik} \left(\frac{\delta_{x_i x_k}}{b_i(x_i)} + \frac{1}{b_i(0)}\right) + \sum_{\alpha \in \mathcal{N}_i} \frac{\partial \lambda_{i\alpha}(x_i)}{\partial b_k(x_k)} \qquad (28)$$

remembering that $b_i(0)$ is a function of $b_i(x_i)$, $x_i \neq 0$. Notice that we need values for $\frac{\partial \lambda_{i\alpha}(x_i)}{\partial b_k(x_k)}$ in order to solve for $\frac{\partial \theta_i(x_i)}{\partial b_k(x_k)}$. Since perturbations in $b_k(x_k)$ (while keeping other $b_j$'s fixed) do not affect nodes not directly connected to $k$, we have $\frac{\partial \lambda_{i\alpha}(x_i)}{\partial b_k(x_k)} = 0$ for $k \notin \alpha$. When $k \in \alpha$, these can in turn be obtained by solving, for each $\alpha$, a matrix inverse. Differentiating (27) by $b_k(x_k)$, we obtain

$$\delta_{ik} \delta_{x_i x_k} = \sum_{j \in \alpha} \sum_{x_j \neq 0} C_{ij}^\alpha(x_i, x_j) \frac{\partial \lambda_{j\alpha}(x_j)}{\partial b_k(x_k)} \qquad (29)$$

$$C_{ij}^\alpha(x_i, x_j) = \begin{cases} b_\alpha(x_i, x_j) - b_i(x_i) b_j(x_j) & \text{if } i \neq j \\ b_i(x_i) \delta_{x_i x_j} - b_i(x_i) b_j(x_j) & \text{if } i = j \end{cases} \qquad (30)$$

for each $i, k \in \mathcal{N}_\alpha$ and $x_i, x_k \neq 0$. Flattening the indices in (29) (varying $i, x_i$ over rows and $k, x_k$ over columns), the LHS becomes the identity matrix, while the RHS is a product

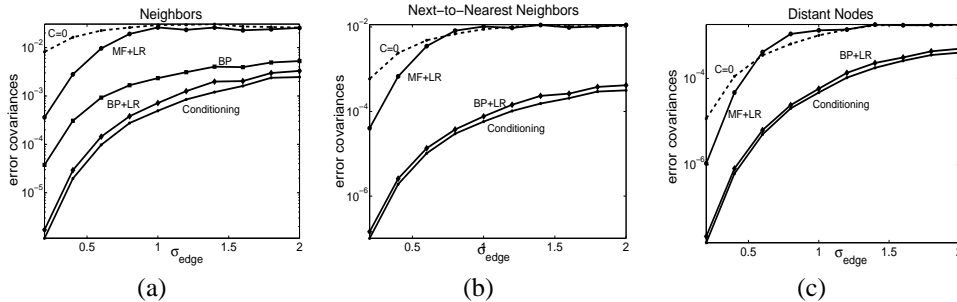

Figure 1: $L_1$-error in covariances for MF+LR, BP, BP+LR and "conditioning". Dashed line is baseline ($C = 0$). The results are separately plotted for neighboring nodes (a), next-to-nearest neighboring nodes (b) and the remaining nodes (c).

of two matrices. The first is a covariance matrix $C_\alpha$ where the $ij^{\text{th}}$ block is $C_{ij}^\alpha(x_i, x_j)$; while the second matrix consists of all the desired derivatives $\frac{\partial \lambda_{j\alpha}(x_j)}{\partial b_k(x_k)}$. Hence the derivatives are given as elements of the inverse covariance matrix $C_\alpha^{-1}$. Finally, plugging the values of $\frac{\partial \lambda_{j\alpha}(x_j)}{\partial b_k(x_k)}$ into (28) now gives $\frac{\partial \theta_i(x_i)}{\partial b_k(x_k)}$ and inverting that matrix will now give us the desired approximate covariances over the whole graph. Interestingly, the method only requires access to the beliefs at the local minimum, not to the potentials or Lagrange multipliers.

## 6   Experiment

The accuracy of the estimated covariances $C_{ij}(x_i, x_j)$ in the LR approximation was studied on a $6 \times 6$ square grid with only nearest neighbors connected and 3 states per node. The solid curves in figure 1 represent the error in the estimates for: 1) mean field + LR approximation [2, 9], 2) BP estimates for neighboring nodes with $b_{\text{EDGE}} = b_\alpha$ in equation (3), 3) BP+LR and 4) "conditioning", where $b_{ij}(x_i, x_j) = b_{i|j}(x_i|x_j) \, b_j^{\text{BP}}(x_j)$ and $b_{i|j}(x_i|x_j)$ is computed by running BP $N \cdot D$ times with $x_j$ clamped at a specific state (this has the same computational complexity as BP+LR). $C$ was computed as $C_{ij} = b_{ij} - b_i b_j$, with $\{b_i, b_j\}$ the marginals of $b_{ij}$, and symmetrizing the result. The error was computed as the absolute difference between the estimated and the true values, averaged over pairs of nodes and their possible states, and averaged over 25 random draws of the network. An instantiation of a network was generated by randomly drawing the logarithm of the edge potentials from a zero mean Gaussian with a standard deviation ranging between $[0, 2]$. The node potentials were set to 1.

From these experiments we conclude that "conditioning" and BP+LR have similar accuracy and significantly outperform MF+LR and BP, while "conditioning" performs slightly better than BP+LR. The latter does however satisfy some desirable properties which are violated by conditioning (see section 7 for further discussion).

## 7   Discussion

In this paper we propose to estimate covariances as follows: first observe that the log partition function is the cumulant generating function, next define its conjugate dual – the Gibbs free energy – and approximate it, finally transform back to obtain a local convex approximation to the log partition function, from which the covariances can be estimated.

The computational complexity of the iterative linear response algorithm scales as $\mathcal{O}(N \cdot$

$E \cdot D^3$) per iteration ($N = \#$nodes, $E = \#$edges, $D = \#$states per node). The non-iterative algorithm scales slightly worse, $\mathcal{O}(N^3 \cdot D^3)$, but is based on a matrix inverse for which very efficient implementations exist. A question that remains open is whether we can improve the efficiency of the iterative algorithm when we are only interested in the joint distributions of *neighboring* nodes.

There are still a number of generalizations worth mentioning. Firstly, the same ideas can be applied to the MF approximation [9] and the Kikuchi approximation (see also [5]). Secondly, the presented method easily generalizes to the computation of higher order cumulants. Thirdly, when applying the same techniques to Gaussian random fields, a propagation algorithm results that computes the inverse of the weight matrix exactly [9]. In the case of more general continuous random field models we are investigating whether linear response algorithms can be applied to the fixed points of expectation propagation.

The most important distinguishing feature between the proposed LR algorithm and the conditioning procedure described in section 6 is the fact that the covariance estimate is automatically positive semi-definite. Indeed the idea to include *global* constraints such as positive semi-definiteness in approximate inference algorithms was proposed in [7]. Other differences include automatic consistency between joint pairwise marginals from LR and node marginals from BP (not true for conditioning) and a convergence proof for the LR algorithm (absent for conditioning, but not observed to be a problem experimentally). Finally, the non-iterative algorithm is applicable to *all* local minima in the Bethe-Gibbs free energy, even those that correspond to unstable fixed points of BP.

### Acknowledgements

We would like to thank Martin Wainwright for discussion. MW would like to thank Geoffrey Hinton for support. YWT would like to thank Mike Jordan for support.

## Footnotes

[1] In extreme cases it is however possible that some entries of $b_{ij}^{\mathrm{LR}}$ become negative.

[2] The unconventional form of this expansion will make subsequent derivations more transparent.

[3]For a more detailed proof of the above two theorems we refer to [9].

# References

[1] T. Heskes. Stable fixed points of loopy belief propagation are minima of the bethe free energy. In *Advances in Neural Information Processing Systems*, volume 15, Vancouver, CA, 2003.

[2] H.J. Kappen and F.B. Rodriguez. Efficient learning in Boltzmann machines using linear response theory. *Neural Computation*, 10:1137–1156, 1998.

[3] F.R. Kschischang, B. Frey, and H.A. Loeliger. Factor graphs and the sum-product algorithm. *IEEE Transactions on Information Theory*, 47(2):498–519, 2001.

[4] M. Opper and O. Winther. From naive mean field theory to the TAP equations. In *Advanced Mean Field Methods – Theory and Practice*. MIT Press, 2001.

[5] K. Tanaka. Probabilistic inference by means of cluster variation method and linear response theory. *IEICE Transactions in Information and Systems*, E86-D(7):1228–1242, 2003.

[6] Y.W. Teh and M. Welling. The unified propagation and scaling algorithm. In *Advances in Neural Information Processing Systems*, 2001.

[7] M.J. Wainwright and M.I. Jordan. Semidefinite relaxations for approximate inference on graphs with cycles. Technical report, Computer Science Division, University of California Berkeley, 2003. Rep. No. UCB/CSD-3-1226.

[8] M. Welling and Y.W. Teh. Approximate inference in boltzmann machines. *Artificial Intelligence*, 143:19–50, 2003.

[9] M. Welling and Y.W. Teh. Linear response algorithms for approximate inference in graphical models. *Neural Computation*, 16:197–221, 2004.

[10] J.S. Yedidia, W. Freeman, and Y. Weiss. Generalized belief propagation. In *Advances in Neural Information Processing Systems*, volume 13, 2000.

[11] A.L. Yuille. CCCP algorithms to minimize the Bethe and Kikuchi free energies: Convergent alternatives to belief propagation. *Neural Computation*, 14(7):1691–1722, 2002.